# ELECTRONIC RECEPTORS FOR TACTILE/HAPTIC* SENSING

Andreas G. Andreou
Electrical and Computer Engineering
The Johns Hopkins University
Baltimore, MD 21218

## ABSTRACT

We discuss synthetic receptors for haptic sensing. These are based on magnetic field sensors (Hall effect structures) fabricated using standard CMOS technologies. These receptors, biased with a small permanent magnet can detect the presence of ferro or ferri-magnetic objects in the vicinity of the sensor. They can also detect the magnitude and direction of the magnetic field.

## INTRODUCTION

The organizational structure and functioning of the sensory periphery in living beings has always been the subject of extensive research. Studies of the retina and the cochlea have revealed a great deal of information as to the ways information is acquired and preprocessed; see for example review chapters in [Barlow and Mollon, 1982]. Understanding of the principles underlying the operation of sensory channels can be utilized to develop machines that can sense their environment and function in it, much like living beings. Although vision is the principal sensory channel to the outside world, the "skin senses" can in some cases provide information that is not available through vision. It is interesting to note, that the performance in identifying objects through the haptic senses can be comparable to vision [Klatzky et. al, 1985]; longer learning periods may be necessary though . Tactually guided exploration and shape perception for robotic applications has been extensively investigated by [Hemami et. al, 1988].

A number of synthetic sensory systems for vision and audition based on physiological models for the retina and the cochlea have been prototyped by Mead and his coworkers in VLSI [Mead, 1989]. The key to success in such endeavors is the ability to integrate transducers (such as light sensitive devices) and local processing electronics on the same chip. A technology that offers that possibility is silicon CMOS; furthermore, it is readily available to engineers and scientists through the MOSIS fabrication services[Cohen and Lewicki, 1981].

Receptor cells, are structures in the sensory pathways whose purpose is to convert environmental signals into electrical activity (strictly speaking this is true for

---

* **Haptic** refers to the perception of vibration, skeletal conformation or position and skin deformation. **Tactile** refers to the perceptual system that includes only the cutaneous senses of vibration and deformation.

exteroceptors). The retina rods and cones are examples of receptors for light stimuli and the Pacinian corpuscles are mechanoreceptors that are sensitive to indentation or pressure on the skin. A *synthetic receptor* is thus the first and necessary functional element in any synthetic sensory system. For the development of vision systems parasitic bipolar devices can be used [Mead, 1985] to perform the necessary light to electrical signal transduction as well as low level signal amplification. On the other hand, implementation of synthetic receptors for tactile perception is still problematic [Barth et. al., 1986]. Truly tactile transducers (devices sensitive to pressure stimuli) are not available in standard CMOS processes and are only found in specialized fabrication lines. However, devices that are sensitive to magnetic fields can be used to some extend as a substitute.

In this paper, we discuss the development of electronic receptor elements that can be used in synthetic haptic/tactile sensing systems. Our receptors are devices which are sensitive to steady state or varying magnetic fields and give electrical signals proportional to the magnetic induction. They can all be fabricated using standard silicon processes such as those offered by MOSIS. We show how our elements can be used for tactile and haptic sensing and compare its characteristics with the features of biological receptors. The spatial resolution of the devices, its frequency response and dynamic range are more than adequate. One of our devices has nano-watt power dissipation and thus can be used in large arrays for high resolution sensing.

## THE MAGNETIC-FIELD SENSORY PARADIGM

In this section we show qualitatively how to implement synthetic sensory functions of the haptic and tactile senses by using magnetic fields and their interaction with ferri or ferro-magnetic objects. This will motivate the more detailed discussion that follows on the transducer devices.

### DIRECT SENSING:

In this mode of operation the transducer will detect the magnitude and direction of the magnetic induction and convert it into an electrical signal. If the magnetic field is provided by the fringing fields of a small permanent magnet, the strength of the signal will fall off with the distance of the sensor from the magnet. Such an arrangement for one dimensional sensing is shown in Figure 1. The experimental data are from the MOS Hall-voltage generator that is described in the next section. The magnetic field was provided by a cylindrical, rare-earth, permanent magnet with magnetic induction Bo=250mT, measured on the end surfaces.The vertical axis shows the signal from the transducer (Hall-voltage) and the horizontal axis represents the distance **d** of the sensor from the surface of the magnet.

The above scheme can be used to sense the angular displacement between two fingers at a joint (inset b). By placing a small magnet on one side of the joint and the receptor on the other, the signal from the receptor can be conditioned and converted into a measure of the angle $\Theta$ between the two fingers. The output of our receptor would thus correspond to the output from the *Joint Fibers* that originate in the *Joint Capsule* [Johnson, 1981]. Joint angle perception and manual stereognosis is mediated in part by these fibers. The above is just one example of how to use our integrated electronic receptor element for sensing

skeletal conformation and position. Since there is no moving parts other than the joint itself, this is a reliable scheme.

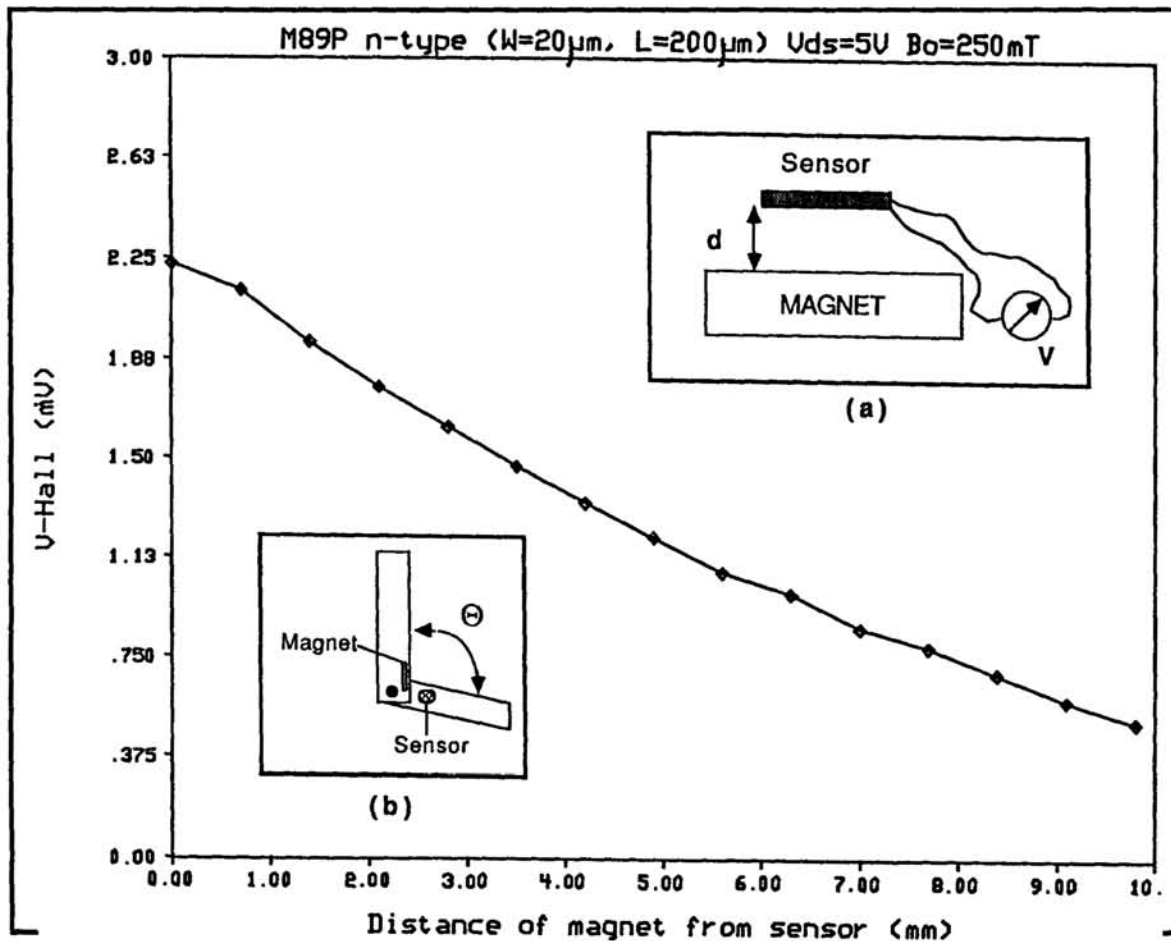

**Figure 1.** Direct Sensing Using an Integrated Hall-Voltage Transducer

## PASSIVE SENSING

In this mode of operation, the device or an array of devices are permanently biased with a uniform magnetic field whose uniformity can be disturbed in the presence of a ferro or ferri-magnetic object in the vicinity. Signals from an array of such elements would provide ifnormation on the shape of the object that causes the disturbance of the magnetic field. Non-magnetic objects can be sensed if the surface of the chip is covered with a compliant membrane that has some magnetic properties. Note that our receptors can detect the presence of an object without having a direct contact with the object itself. This may in some cases be advantageous.

In this application, the magnetic field sensor would act more like the Ruffini organs which exist in the deeper tissue and are primarily sensitive to static skin stretch. The above scheme could also be used for sensing dynamic stimuli and there is a variety of receptor cells, such as the *Pacinian and Meissner's corpuscles* that perform that function in biological tactile senses [Johnson, 1981].

## SILICON TRANSDUCERS

Magnetic field sensors can be integrated on silicon in a variety of forms. The transduction mechanism is due to some galvanomagnetic effect; the Hall effect or some related phenomenon [Andreou, 1986]. For an up-todate review of integrated magnetic field sensors as well as for the fine points of the discussion that follows in the next two sections, please refer to [Baltes and Popovic, 1986]. The simplest Hall device is the Hall-voltage sensor. This is a four terminal device with two current terminals and two voltage terminals to measure the Hall-voltage (Figure 2). A magnetic field **B** in the direction perpendicular to the current flow, sets up the Hall-voltage in the direction indicated in the figure. The Hall-voltage is such that it compensates for the Lorentz force on the charge carriers. In the experiment below, we have used a MOS Hall generator instead of a bulk device. The two current contacts are the source and drain of the device and the voltage contacts are formed by small diffusion areas in the channel. The Hall-voltage is linearly related to the magnetic induction **B** and the total current in the sample **I** between the drain and the source which is controlled by the gate voltage Vgs.

$$V_H = KBI$$

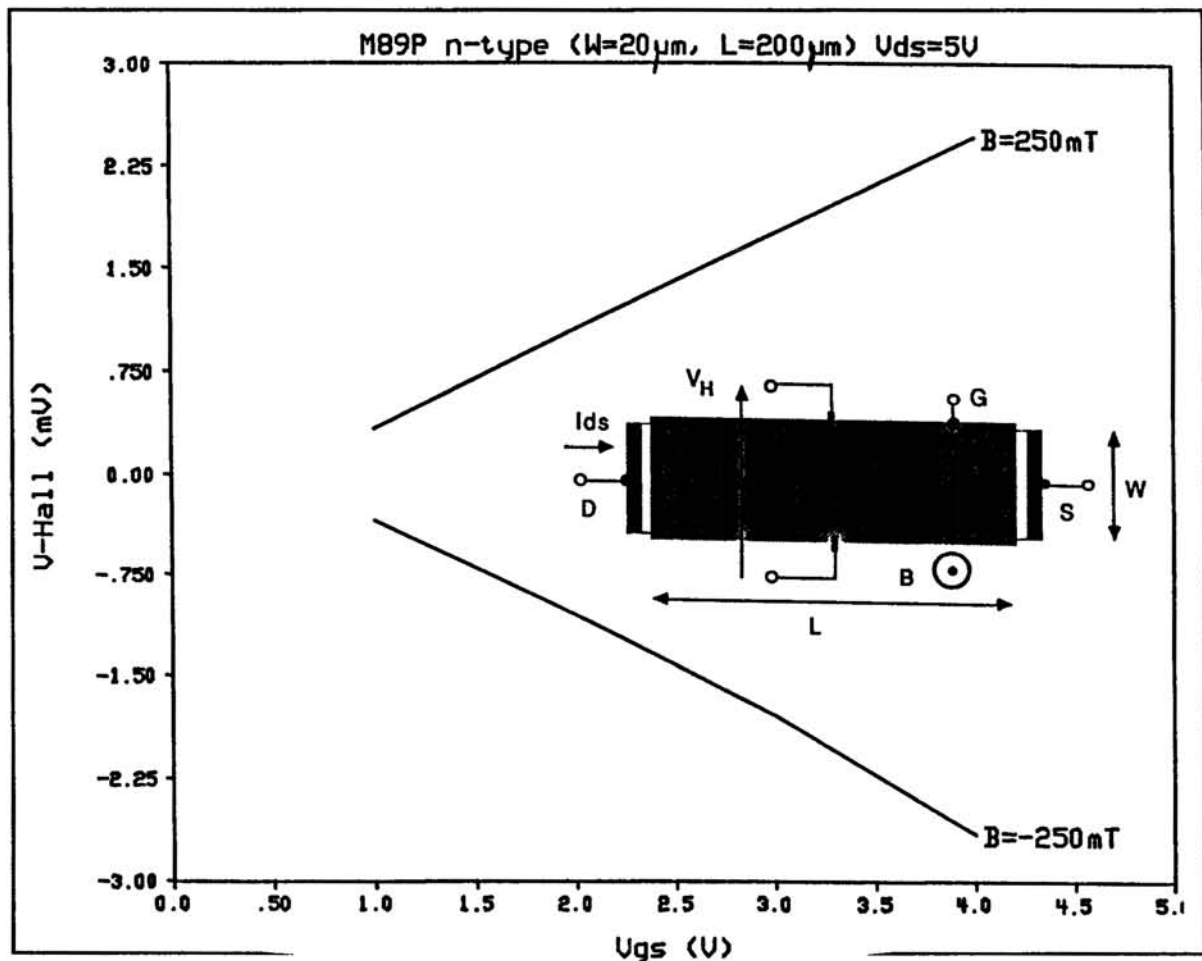

**Figure 2.** An Integrated MOS Hall-Voltage Sensor and Measured Characteristics

The constant of proportionality **K** is related to the dimensions of the device and to silicon electronic transport parameters. The device dimensions and biasing conditions are shown in the figure above. Note that the Hall voltage reverses, when the direction of the magnetic field is reversed. The above device was fabricated in a standard 2-micron n-well CMOS process through MOSIS (production run M89P). The signal output of this sensor is a voltage and is relatively small. For the same biasing conditions, the signals can be increased only if the channel is made shorter (increase the transconductance of the device). On the otherhand, when the length of the device approaches its width the Hall-voltage is shorted out by the heavily doped source and drain regions and the signal degrades again. Some of the problems with the Hall-voltage sensor can be avoided if we use a device that gives a current as its output signal; this is discussed in the next section.

## THE HALL-CURRENT* SENSOR

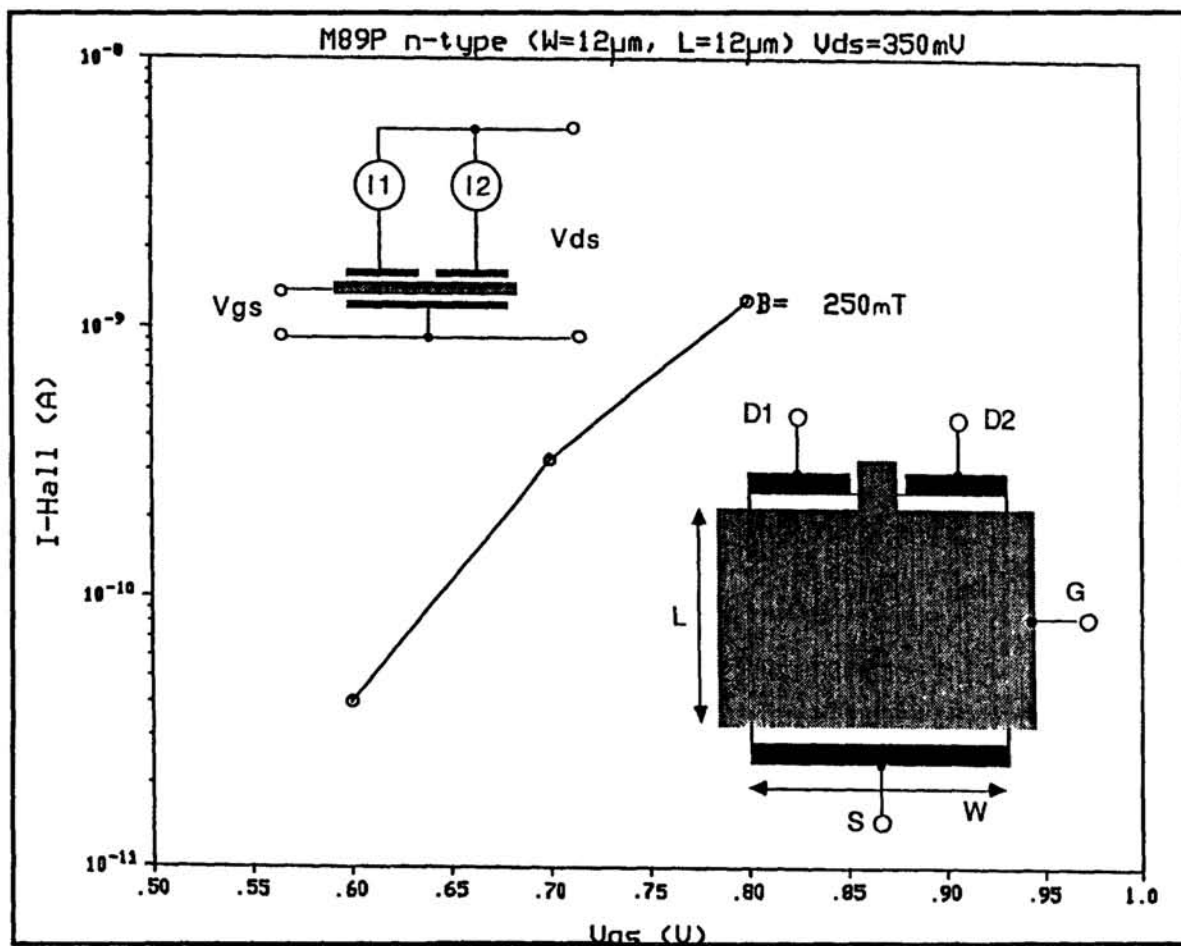

**Figure 3.** The Hall-Current Sensor

This current is a direct consequence of the Lorentz force and is perpendicular to the direction of current flow without a magnetic field. The Lorentz force:

$$\hat{F} = q\vec{u} \times \hat{B}$$

depends on the velocity of the carriers in the sample and on the magnetic induction. Since this force is responsible for the transverse current flow a more appropriate name for this sensor is a *Lorentz-current* sensor. Obviously, given a magnetic field strength, if we want a maximum signal from our device we want the carriers to have the maximum velocity in the channel. We achieve that by operating the devices in the saturation region were the carriers transverse the channel at what is called the "saturation velocity". In this configuration we can also use short channel devices (and consequently smaller devices) so that the high fields in the channel can be set with lower voltages. The geometry for such sensor as well as the biasing circuit is shown in the insets of Figure 3. As with the previous sensor, the magnetic field is applied in the direction perpendicular to the channel (also perpendicular to the plane of the gate electrode).

This sensor is a MOS transistor with three terminals and the gate. It has a single source but a split drain. The polysilicon gate is extended in the region between the two drains so that they are electrically independent of each other. The device is biased with a constant drain-source voltage (with the two drains at the same potential) and the currents from the two sources are monitored. The current-mode circuitry for the synthetic neurons described by Kwabena [Kwabena et. al., 1989] can be employed for this function. We operate the device in the subthreshold region (denoted by the gate-source volrages between 0.5 and 0.9 volts). On the application of a transverse magnetic field we observe the imbalance between the two drain currents. The Hall-current, plotted in Figure 3, is twice that current.

$$I_{Hall} = 2\left| I_2 - I_1 \right| = K_h I_{ds} B$$

Note that we can operate the device and observe an effect due to the magnetic field at very low currents in the nano-amp range. The graph in Figure 3 shows also the dependence of the Hall-current on the gate voltage. This is a logarithmic relationship because the Hall-current is directly related to the total current in the sample **Ids** through a linear relationship; it is also linearly related to the magnetic field with a proportionality constant **Kh**. The derivation of a formula for the Hall-current can be found in [Andreou, 1986].

## DISCUSSION

The frequency response of the sensors described above is more than adequate for our applications. The frequency response of the Hall effect mechanism is many order of magnitude higher than the requirements of our circuits which have to work only the Hz an kHz range. Another important criterion for the receptors is their spatial resolution. We have fabricated and tested devices with areas as small as 144 square microns. These are comparable or even better with what is found in biological systems (10 to 150 receptors per square cm). On the otherhand, it is more likely that our final "receptor" elements will be larger, partly, because of the additional electronics. The experimental data shown above are only for stimuli that are static and are simply the raw output from our

transducer itself. Clearly such output signals will not be of much use without some local processing. For example, *adaptation mechanisms* have to be included in our synthetic receptors for cutaneous sensing. The sensitivity of our transducer elements may be a problem. In that case more sophisticated structures such as parasitic bipolar magnetotransistors or combination of MOS Hall and bipolar devices can be employed for low level signal amplification [Andreou, 1986]. Voltage offsets in the Hall-voltage sensor would also present some problem; the same is true for current imbalances due to fabrication imperfections in the Hall current sensor. One of the most attractive properties of the Hall-current type sensor described in this paper is its ability to work with very lower voltages and very low currents; one of our devices can operate with bias voltage as low as 350mV and total current of 1nA without compromising its sensitivity. Power dissipation may be a problem when large arrays of these devices are considered.

Devices for sensing temperature can also be implemented on a standard silicon CMOS process. Thus a multisensor chip, could be designed that would respond to more than one of the somatic senses.

## CONCLUSIONS

We have demonstrated how to use the magnetic field as a paradigm for haptic sensing. We have also reported on a silicon magnetic field sensor that operates with power dissipation as low as 350pW without any compromise in its performance. This is a dual-drain MOS Hall-current device operating in the subthreshold region. Our elements are only very simple "receptors" without any adaptation mechanisms or local processing; this will be the next step in our work.

## Acknowledg ments

This work was funded by the Independent Research and Development program of the Johns Hopkins University, Applied Physics Laboratory. The support and personal interest of Robert Jenkins is gratefully acknowledged. The author has benefited by the occasional contact with Ken Johnson of the Biomedical Engineering Department.

## Footnotes

* Hall current is a misnomer (used also by this author) that exists in the literature to characterize the current flow in the sample when the Hall field is shorted out.

## References

H.B. Barlow and J.D. Mollon eds.,*The senses*, Cambridge University Press, Oxford, 1982.

R.L. Klatzky, S.J. Lederman and V.A. Metzger, "Identifying objects by touch: An `expert system'," *Percept. Psychophys.* vol. 37, 1985.

H. Hemami, J.S. Bay and R.E. Goddard, "A Conceptual Framework for Tactually Guided Exploration and Shape Perception," *IEEE Trans. Biomedical Engineering*, vol. 35, No. 2, Feb. 1988.

C.A. Mead, *Analog VLSI and Neural Systems,* Addison and Wesley, (in press).

D. Cohen and G. Lewicki, "MOSIS-The ARPA silicon broker," *Proc. of the Second Caltech Conference on VLSI,* Pasadena, California,1981.

C.A. Mead, A sensitive electronic photoreptor, *1985 Chapel Hill Conference on VLSI,* Chapel Hill, 1985.

P.W. Barth, M.J. Zdeblick, Z. Kuc and P.A. Beck, "Flexible tactile arrays for robotics: architectural robustness and yield considerations, *Tech. Digest, IEEE Solid State Sensors Workshop*, Hilton Head Island, 1986.

A.G. Andreou, "*The Hall effect and related phenomena in microelectronic devices.*" Ph.D. Dissertation, The Johns Hopkins University, Baltimore, MD, 1986.

H.P. Baltes and R.S. Popovic, "Integrated Semiconductor Magnetic Field Sensors," *Proceedings IEEE*, vol.74, No. 8, Aug. 1986.

K.O. Johnson and G.D. Lamb, "Neural mechanisms of spatial discrimination: Neural patterns evoked by Braille-like dot patterns in the monkey," *J. of Phys.*, vol. 310, pp. 117-144, 1981.

K.A. Boahen, A.G. Andreou, P.O. Pouliquen and A. Pavasovic, "Architectures for Associative Memories Using Current-Mode Analog MOS Circuits," *Proceedings of the Decennial Caltech Conference on VLSI*, C. Seitz ed. MIT Press, 1989.

